# GTM: A Principled Alternative to the Self-Organizing Map

**Christopher M. Bishop**
C.M.Bishop@aston.ac.uk

**Markus Svensén**
svensjfm@aston.ac.uk

**Christopher K. I. Williams**
C.K.I.Williams@aston.ac.uk

Neural Computing Research Group
Aston University, Birmingham, B4 7ET, UK
http://www.ncrg.aston.ac.uk/

## Abstract

The Self-Organizing Map (SOM) algorithm has been extensively studied and has been applied with considerable success to a wide variety of problems. However, the algorithm is derived from heuristic ideas and this leads to a number of significant limitations. In this paper, we consider the problem of modelling the probability density of data in a space of several dimensions in terms of a smaller number of latent, or hidden, variables. We introduce a novel form of latent variable model, which we call the GTM algorithm (for *Generative Topographic Mapping*), which allows general non-linear transformations from latent space to data space, and which is trained using the EM (expectation-maximization) algorithm. Our approach overcomes the limitations of the SOM, while introducing no significant disadvantages. We demonstrate the performance of the GTM algorithm on simulated data from flow diagnostics for a multi-phase oil pipeline.

## 1 Introduction

The Self-Organizing Map (SOM) algorithm of Kohonen (1982) represents a form of unsupervised learning in which a set of unlabelled data vectors $t_n$ ($n = 1, \ldots, N$) in a $D$-dimensional data space is summarized in terms of a set of reference vectors having a spatial organization corresponding (generally) to a two-dimensional sheet[1].

While this algorithm has achieved many successes in practical applications, it also suffers from some major deficiencies, many of which are highlighted in Kohonen (1995) and reviewed in this paper.

From the perspective of statistical pattern recognition, a fundamental goal in unsupervised learning is to develop a representation of the distribution $p(\mathbf{t})$ from which the data were generated. In this paper we consider the problem of modelling $p(\mathbf{t})$ in terms of a number (usually two) of *latent* or *hidden* variables. By considering a particular class of such models we arrive at a formulation in terms of a constrained Gaussian mixture which can be trained using the EM (expectation-maximization) algorithm. The topographic nature of the representation is an intrinsic feature of the model and is not dependent on the details of the learning process. Our model defines a *generative* distribution $p(\mathbf{t})$ and will be referred to as the GTM (*Generative Topographic Mapping*) algorithm (Bishop *et al.*, 1996a).

## 2   Latent Variables

The goal of a latent variable model is to find a representation for the distribution $p(\mathbf{t})$ of data in a $D$-dimensional space $\mathbf{t} = (t_1, \ldots, t_D)$ in terms of a number $L$ of latent variables $\mathbf{x} = (x_1, \ldots, x_L)$. This is achieved by first considering a non-linear function $\mathbf{y}(\mathbf{x}; \mathbf{W})$, governed by a set of parameters $\mathbf{W}$, which maps points $\mathbf{x}$ in the latent space into corresponding points $\mathbf{y}(\mathbf{x}; \mathbf{W})$ in the data space. Typically we are interested in the situation in which the dimensionality $L$ of the latent space is less than the dimensionality $D$ of the data space, since our premise is that the data itself has an intrinsic dimensionality which is less than $D$. The transformation $\mathbf{y}(\mathbf{x}; \mathbf{W})$ then maps the latent space into an $L$-dimensional non-Euclidean manifold embedded within the data space.

If we define a probability distribution $p(\mathbf{x})$ on the latent space, this will induce a corresponding distribution $p(\mathbf{y}|\mathbf{W})$ in the data space. We shall refer to $p(\mathbf{x})$ as the prior distribution of $\mathbf{x}$ for reasons which will become clear shortly. Since $L < D$, the distribution in $\mathbf{t}$-space would be confined to a manifold of dimension $L$ and hence would be singular. Since in reality the data will only approximately live on a lower-dimensional manifold, it is appropriate to include a noise model for the $\mathbf{t}$ vector. We therefore define the distribution of $\mathbf{t}$, for given $\mathbf{x}$ and $\mathbf{W}$, to be a spherical Gaussian centred on $\mathbf{y}(\mathbf{x}; \mathbf{W})$ having variance $\beta^{-1}$ so that $p(\mathbf{t}|\mathbf{x}, \mathbf{W}, \beta) \sim \mathcal{N}(\mathbf{t}|\mathbf{y}(\mathbf{x}; \mathbf{W}), \beta^{-1}\mathbf{I})$. The distribution in $\mathbf{t}$-space, for a given value of $\mathbf{W}$, is then obtained by integration over the $\mathbf{x}$-distribution

$$p(\mathbf{t}|\mathbf{W}, \beta) = \int p(\mathbf{t}|\mathbf{x}, \mathbf{W}, \beta) p(\mathbf{x}) \, d\mathbf{x}. \tag{1}$$

For a given a data set $\mathcal{D} = (\mathbf{t}_1, \ldots, \mathbf{t}_N)$ of $N$ data points, we can determine the parameter matrix $\mathbf{W}$, and the inverse variance $\beta$, using maximum likelihood, where the log likelihood function is given by

$$L(\mathbf{W}, \beta) = \sum_{n=1}^{N} \ln p(\mathbf{t}_n|\mathbf{W}, \beta). \tag{2}$$

In principle we can now seek the maximum likelihood solution for the weight matrix, once we have specified the prior distribution $p(\mathbf{x})$ and the functional form of the

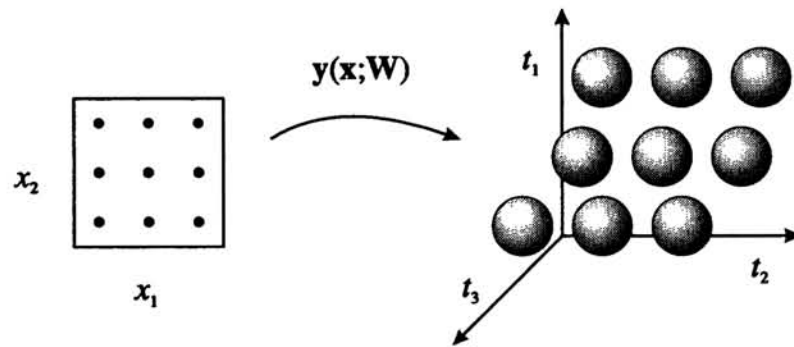

Figure 1: We consider a prior distribution $p(\mathbf{x})$ consisting of a superposition of delta functions, located at the nodes of a regular grid in latent space. Each node $\mathbf{x}_l$ is mapped to a point $\mathbf{y}(\mathbf{x}_l; \mathbf{W})$ in data space, which forms the centre of the corresponding Gaussian distribution.

mapping $\mathbf{y}(\mathbf{x}; \mathbf{W})$, by maximizing $L(\mathbf{W}, \beta)$. The latent variable model can be related to the Kohonen SOM algorithm by choosing $p(\mathbf{x})$ to be a sum of delta functions centred on the nodes of a regular grid in latent space $p(\mathbf{x}) = 1/K \sum_{l=1}^{K} \delta(\mathbf{x} - \mathbf{x}_l)$. This form of $p(\mathbf{x})$ allows the integral in (1) to be performed analytically. Each point $\mathbf{x}_l$ is then mapped to a corresponding point $\mathbf{y}(\mathbf{x}_l; \mathbf{W})$ in data space, which forms the centre of a Gaussian density function, as illustrated in Figure 1. Thus the distribution function in data space takes the form of a Gaussian mixture model $p(\mathbf{t}|\mathbf{W}, \beta) = 1/K \sum_{l=1}^{K} p(\mathbf{t}|\mathbf{x}_l, \mathbf{W}, \beta)$ and the log likelihood function (2) becomes

$$L(\mathbf{W}, \beta) = \sum_{n=1}^{N} \ln \left\{ \frac{1}{K} \sum_{l=1}^{K} p(\mathbf{t}_n|\mathbf{x}_l, \mathbf{W}, \beta) \right\}. \tag{3}$$

This distribution is a *constrained* Gaussian mixture since the centres of the Gaussians cannot move independently but are related through the function $\mathbf{y}(\mathbf{x}; \mathbf{W})$. Note that, provided the mapping function $\mathbf{y}(\mathbf{x}; \mathbf{W})$ is smooth and continuous, the projected points $\mathbf{y}(\mathbf{x}_l; \mathbf{W})$ will necessarily have a topographic ordering.

## 2.1   The EM Algorithm

If we choose a particular parametrized form for $\mathbf{y}(\mathbf{x}; \mathbf{W})$ which is a differentiable function of $\mathbf{W}$ we can use standard techniques for non-linear optimization, such as conjugate gradients or quasi-Newton methods, to find a weight matrix $\mathbf{W}^*$, and inverse variance $\beta^*$, which maximize $L(\mathbf{W}, \beta)$. However, our model consists of a mixture distribution which suggests that we might seek an EM algorithm (Dempster *et al.*, 1977). By making a careful choice of model $\mathbf{y}(\mathbf{x}; \mathbf{W})$ we will see that the M-step can be solved exactly. In particular we shall choose $\mathbf{y}(\mathbf{x}; \mathbf{W})$ to be given by a generalized linear network model of the form

$$\mathbf{y}(\mathbf{x}; \mathbf{W}) = \mathbf{W}\boldsymbol{\phi}(\mathbf{x}) \tag{4}$$

where the elements of $\boldsymbol{\phi}(\mathbf{x})$ consist of $M$ fixed basis functions $\phi_j(\mathbf{x})$, and $\mathbf{W}$ is a $D \times M$ matrix with elements $w_{kj}$. Generalized linear networks possess the same universal approximation capabilities as multi-layer adaptive networks, provided the basis functions $\phi_j(\mathbf{x})$ are chosen appropriately.

By setting the derivatives of (3) with respect to $w_{kj}$ to zero, we obtain

$$\mathbf{\Phi}^{\mathrm{T}}\mathbf{G}\mathbf{\Phi}\mathbf{W}^{\mathrm{T}} = \mathbf{\Phi}^{\mathrm{T}}\mathbf{R}\mathbf{T} \tag{5}$$

where $\mathbf{\Phi}$ is a $K \times M$ matrix with elements $\Phi_{lj} = \phi_j(\mathbf{x}_l)$, $\mathbf{T}$ is a $N \times D$ matrix with elements $t_{kn}$, and $\mathbf{R}$ is a $K \times N$ matrix with elements $R_{ln}$ given by

$$R_{ln}(\mathbf{W}, \beta) = \frac{p(\mathbf{t}_n|\mathbf{x}_l, \mathbf{W}, \beta)}{\sum_{l'=1}^{K} p(\mathbf{t}_n|\mathbf{x}_{l'}, \mathbf{W}, \beta)} \tag{6}$$

which represents the posterior probability, or *responsibility*, of the mixture components $l$ for the data point $n$. Finally, $\mathbf{G}$ is a $K \times K$ diagonal matrix, with elements $G_{ll} = \sum_{n=1}^{N} R_{ln}(\mathbf{W}, \beta)$. Equation (5) can be solved for $\mathbf{W}$ using standard matrix inversion techniques. Similarly, optimizing with respect to $\beta$ we obtain

$$\frac{1}{\beta} = \frac{1}{ND}\sum_{l=1}^{K}\sum_{n=1}^{N} R_{ln}(\mathbf{W}, \beta)\|\mathbf{y}(\mathbf{x}_l; \mathbf{W}) - \mathbf{t}_n\|^2. \tag{7}$$

Here (6) corresponds to the E-step, while (5) and (7) correspond to the M-step. Typically the EM algorithm gives satisfactory convergence after a few tens of cycles. An on-line version of this algorithm can be obtained by using the Robbins-Monro procedure to find a zero of the objective function gradient, or by using an on-line version of the EM algorithm.

## 3   Relation to the Self-Organizing Map

The list below describes some of the problems with the SOM procedure and how the GTM algorithm solves them.

1. The SOM algorithm is not derived by optimizing an objective function, unlike GTM. Indeed it has been proven (Erwin *et al.*, 1992) that such an objective function cannot exist for the SOM algorithm.

2. In GTM the neighbourhood-preserving nature of the mapping is an automatic consequence of the choice of a smooth, continuous function $\mathbf{y}(\mathbf{x}; \mathbf{W})$. Neighbourhood-preservation is not guaranteed by the SOM procedure.

3. There is no assurance that the code-book vectors will converge using SOM. Convergence of the batch GTM algorithm is guaranteed by the EM algorithm, and the Robbins-Monro theorem provides a convergence proof for the on-line version.

4. GTM defines an explicit probability density function in data space. In contrast, SOM does not define a density model. Attempts have been made to interpret the density of codebook vectors as a model of the data distribution but with limited success. The advantages of having a density model include the ability to deal with missing data in a principled way, and the straightforward possibility of using a mixture of such models, again trained using EM.

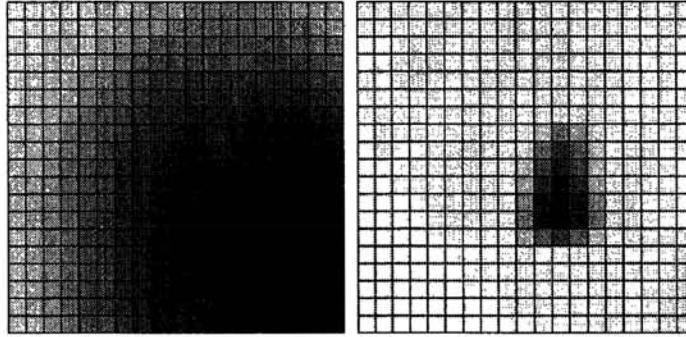

Figure 2:   Examples of the posterior probabilities (responsibilities) of the latent space points at an early stage (left) and late stage (right) during the convergence of the GTM algorithm, evaluated for a single data point from the training set in the oil-flow problem discussed in Section 4. Note how the probabilities form a localized 'bubble' whose size shrinks *automatically* during training, in contrast to the hand-crafted shrinkage of the neighbourhood function in the SOM.

5. For SOM the choice of how the neighbourhood function should shrink over time during training is arbitrary, and so this must be optimized empirically. There is no neighbourhood function to select for GTM.

6. It is difficult to know by what criteria to compare different runs of the SOM procedure. For GTM one simply compares the likelihood of the data under the model, and standard statistical tests can be used for model comparison.

Notwithstanding these key differences, there are very close similarities between the SOM and GTM techniques. Figure 2 shows the posterior probabilities (responsibilities) corresponding to the oil flow problem considered in Section 4. At an early stage of training the responsibility for representing a particular data point is spread over a relatively large region of the map. As the EM algorithm proceeds so this responsibility 'bubble' shrinks automatically. The responsibilities (computed in the E-step) govern the updating of $\mathbf{W}$ and $\beta$ in the M-step and, together with the smoothing effect of the basis functions $\phi_j(\mathbf{x})$, play an analogous role to the neighbourhood function in the SOM procedure. While the SOM neighbourhood function is arbitrary, however, the shrinking responsibility bubble in GTM arises directly from the EM algorithm.

## 4   Experimental Results

We present results from the application of this algorithm to a problem involving 12-dimensional data arising from diagnostic measurements of oil flows along multi-phase pipelines (Bishop and James, 1993). The three phases in the pipe (oil, water and gas) can belong to one of three different geometrical configurations, corresponding to stratified, homogeneous, and annular flows, and the data set consists of 1000 points drawn with equal probability from the 3 classes. We take the latent variable space to be two-dimensional, since our goal in this application is data visualization. Each data point $\mathbf{t}_n$ induces a posterior distribution $p(\mathbf{x}|\mathbf{t}_n, \mathbf{W}, \beta)$ in x-space. However, it is often convenient to project each data point down to a unique point in x-space, which can be done by finding the mean of the posterior distribution.

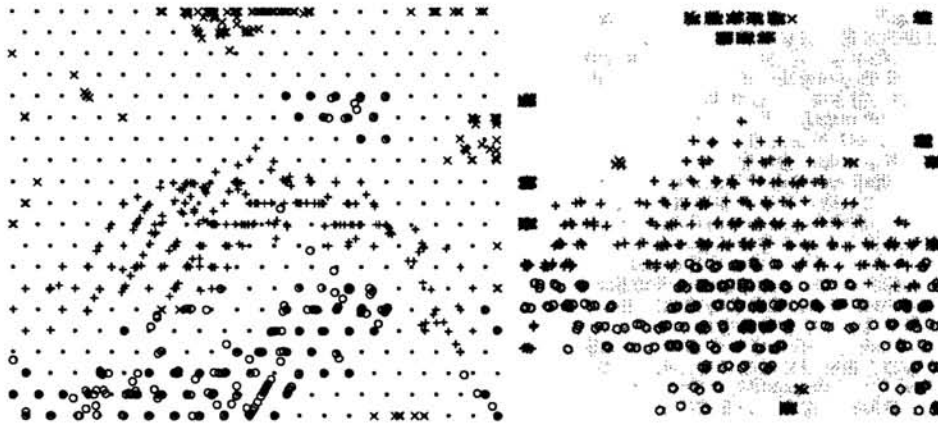

Figure 3: The left plot shows the posterior-mean projection of the oil flow data in the latent space of the non-linear model. The plot on the right shows the same data set visualized using the batch SOM procedure, in which each data point is assigned to the point on the feature map corresponding to the codebook vector to which it is nearest. In both plots, crosses, circles and plus-signs represent the three different oil-flow configurations.

Figure 3 shows the oil data visualized with GTM and SOM. The CPU times taken for the GTM, SOM with a Gaussian neighbourhood, and SOM with a 'top-hat' neighbourhood were 644, 1116 and 355 seconds respectively. In each case the algorithms were run for 25 complete passes through the data set.

## 5  Discussion

In the fifteen years since the SOM procedure was first proposed, it has been used with great success in a wide variety of applications. It is, however, based on heuristic concepts, rather than statistical principles, and this leads to a number of serious deficiencies in the algorithm. There have been several attempts to provide algorithms which are similar in spirit to the SOM but which overcome its limitations, and it is useful to compare these to the GTM algorithm.

The formulation of the elastic net algorithm described by Durbin *et al.* (1989) also constitutes a Gaussian mixture model in which the Gaussian centres acquire a spatial ordering during training. The principal difference compared with GTM is that in the elastic net the centres are independent parameters but are encouraged to be spatially close by the use of a quadratic regularization term, whereas in GTM there is no regularizer on the centres and instead the centres are constrained to lie on a manifold given by the non-linear projection of the latent-variable space. The existence of a well-defined manifold means that the local magnification factors can be evaluated explicitly as continuous functions of the latent variables (Bishop *et al.*, 1996b). By contrast, in algorithms such as SOM and the elastic net, the embedded manifold is defined only indirectly as a discrete approximation by the locations of the code-book vectors or Gaussian centres.

One version of the principal curves algorithm (Tibshirani, 1992) introduces a generative distribution based on a mixture of Gaussians, with a well-defined likelihood function, which is trained by the EM algorithm. However, the number of Gaussian components is equal to the number of data points, and the algorithm has been

formulated for one-dimensional manifolds, making this algorithm further removed from the SOM.

MacKay (1995) considers convolutional models of the form (1) using multi-layer network models in which a discrete sample from the latent space is interpreted as a Monte Carlo approximation to the integration over a continuous distribution. Although an EM approach could be applied to such a model, the M-step of the corresponding EM algorithm would itself require a non-linear optimization.

In conclusion, we have provided an alternative algorithm to the SOM which overcomes its principal deficiencies while retaining its general characteristics. We know of no significant disadvantage in using the GTM algorithm in place of the SOM. While we believe the SOM procedure is superseded by the GTM algorithm, is should be noted that the SOM has provided much of the inspiration for developing GTM.

A web site for GTM is provided at:

$$\texttt{http://www.ncrg.aston.ac.uk/GTM/}$$

which includes postscript files of relevant papers, software implementations in Matlab and C, and example data sets used in the development of the GTM algorithm.

### Acknowledgements

This work was supported by EPSRC grant GR/K51808: *Neural Networks for Visualisation of High-Dimensional Data*. Markus Svensén would like to thank the staff of the SANS group in Stockholm for their hospitality during part of this project.

## Footnotes

[1] Biological metaphor is sometimes invoked when motivating the SOM procedure. It should be stressed that our goal here is not neuro-biological modelling, but rather the development of effective algorithms for data analysis.

# References

Bishop, C. M. and G. D. James (1993). Analysis of multiphase flows using dual-energy gamma densitometry and neural networks. *Nuclear Instruments and Methods in Physics Research* **A327**, 580–593.

Bishop, C. M., M. Svensén, and C. K. I. Williams (1996a). Gtm: The generative topographic mapping. Technical Report NCRG/96/015, Neural Computing Research Group, Aston University, Birmingham, UK. Submitted to Neural Computation.

Bishop, C. M., M. Svensén, and C. K. I. Williams (1996b). Magnification factors for the GTM algorithm. In preparation.

Dempster, A. P., N. M. Laird, and D. B. Rubin (1977). Maximum likelihood from incomplete data via the EM algorithm. *Journal of the Royal Statistical Society, B* **39** (1), 1–38.

Durbin, R., R. Szeliski, and A. Yuille (1989). An analysis of the elastic net approach to the travelling salesman problem. *Neural Computation* **1** (3), 348–358.

Erwin, E., K. Obermayer, and K. Schulten (1992). Self-organizing maps: ordering, convergence properties and energy functions. *Biological Cybernetics* **67**, 47–55.

Kohonen, T. (1982). Self-organized formation of topologically correct feature maps. *Biological Cybernetics* **43**, 59–69.

Kohonen, T. (1995). *Self-Organizing Maps*. Berlin: Springer-Verlag.

MacKay, D. J. C. (1995). Bayesian neural networks and density networks. *Nuclear Instruments and Methods in Physics Research, A* **354** (1), 73–80.

Tibshirani, R. (1992). Principal curves revisited. *Statistics and Computing* **2**, 183–190.